# Functional Models of Selective Attention and Context Dependency

**Thomas H. Hildebrandt**
Department of Electrical Engineering and Computer Science
Room 304 Packard Laboratory          19 Memorial Drive West
Lehigh University          Bethlehem PA  18015-3084
thildebr@aragorn.eecs.lehigh.edu

## Scope

This workshop reviewed and classified the various models which have emerged from the general concept of selective attention and context dependency, and sought to identify their commonalities. It was concluded that the motivation and mechanism of these functional models are "efficiency" and "factoring", respectively. The workshop focused on computational models of selective attention and context dependency within the realm of neural networks. We treated only "functional" models; computational models of biological neural systems, and symbolic or rule-based systems were omitted from the discussion.

## Presentations

**Thomas H. Hildebrandt** presented the results of his recent survey of the literature on functional models of selective attention and context dependency. He set forth the notions that selective attention and context dependency are equivalent, that the goal of these methods is to reduce computational requirements, and that this goal is achieved by what amounts to factoring or a divide-and-conquer technique which takes advantage of nonlinearities in the problem.

**Daniel S. Levine** (University of Texas at Arlington) showed how the gated dipole structure often used in the ART models can be used to account for time-dependent phenomena such as habituation and overcompensation. His adjusted model appropriately modelled the public's adverse reaction to "New Coke".

**Lev Goldfarb** (University of New Brunswick) presented a formal model for inductive learning based on *symbolic* transformation systems and *parametric* distance functions as an alternative to the commonly used *algebraic* transformation system and *Euclidean* distance function. The drawbacks of the latter system were briefly discussed, and it was shown how this new formal system can give rise to learning models which overcome these problems.

**Chalapathy Neti** (IBM, Boca Raton) presented a model which he has used to increase signal-to-noise ratio (SNR) in noisy speech signals. The model is based on adaptive filtering of frequency bands with a constant frequency to bandwidth ratio. This thresholding in the wavelet domain gives results which are superior to similar methods operating in the Adaptive Fourier domain. Several types of signal could be detected with SNRs close to 0db.

**Paul N. Refenes** (University of London Business School) demonstrated the need to take advantage of contextual information in attempting to model the capital markets. There exist some fundamental economic formulae, but they hold only in the long term. The desire to model events on a finer time scale requires reference to significant factors within a smaller window. To do this effectively requires the identification of appropriate short-term indicators, as mere overparameterization has been shown to lead to negative results.

**Jonathan A. Marshall** (University of North Carolina) reviewed the EXIN model, which correctly encodes partially overlapping patterns as distinct activations in the output layer, while allowing the simultaneous appearance of nonoverlapping patterns to give rise to multiple activations in the output layer. The model thus produces a factored representation of complex scenes.

**Albert Nigrin** (American University) presented a model, similar in concept to the EXIN model. It correctly handles synonymous inputs by means of cross-inhibition of the links connecting the synonyms to the target node.

**Thomas H. Hildebrandt** also presented a model for adaptive classification based on decision feedback equalization. The model shifts the decision boundaries of the underlying classifier to compensate shifts in the statistics of the input. On handwritten character classification, it outperformed an identical classifier which used only static decision boundaries.

## Summary

According to Hildebrandt's first talk, the concepts underlying selective attention are quite broad and generally applicable. Large nonlinearities in the problem permit the use of problem subdivision or factoring (by analogy with the factoring of a Boolean equation). Factoring is a good method for reducing the complexity of nonlinear systems.

The talks by Levine and Refenes showed that context enters naturally into the description, formulation, and solution of real-world modelling problems. Those by Neti and Hildebrandt showed that specific reference to temporal context can result in immediate performance gains. The presentations by Marshall and Nigrin provided models for appropriately encoding contexts involving overlapping and synonymous patterns, respectively. The talk by Goldfarb indicates that abandoning assumptions regarding linearity *ab initio* may lead to more powerful learning systems. Refer to [1] for further information.

## References

[1] Hildebrandt, Thomas H. Neural Network Models for Selective Attention and Context Dependency. Submitted to *Neural Networks*, December 1993.
